# Interactive Parts Model: an Application to Recognition of On-line Cursive Script

**Predrag Neskovic, Philip C Davis\* and Leon N Cooper**
Physics Department and Institute for Brain and Neural Systems
Brown University, Providence, RI 02912

## Abstract

In this work, we introduce an Interactive Parts (IP) model as an alternative to Hidden Markov Models (HMMs). We tested both models on a database of on-line cursive script. We show that implementations of HMMs and the IP model, in which all letters are assumed to have the same average width, give comparable results. However, in contrast to HMMs, the IP model can handle duration modeling without an increase in computational complexity.

## 1  Introduction

Hidden Markov models [9] have been a dominant paradigm in speech and handwriting recognition over the past several decades. The success of HMMs is primarily due to their ability to model the statistical and sequential nature of speech and handwriting data. However, HMMs have a number of weaknesses [2]. First, discriminative powers of HMMs are weak since the training algorithm is based on a Maximum Likelihood Estimate (MLE) criterion, whereas the optimal training should be based on a Maximum a Posteriori (MAP) criterion [2]. Second, in most HMMs, only first or second order dependencies are assumed. Although explicit duration HMMs model data more accurately, the computational cost of such modeling is high [5].

To overcome the first problem, it has been suggested [1, 11, 2] that Neural Networks (NNs) should be used for estimating emission probabilities. Since NNs cannot deal well with sequential data, they are often used in combination with HMMs as hybrid NN/HMM systems [2, 11].

In this work, we introduce a new model that provides a possible solution to the second problem. In addition, this new objective function can be cast into a NN-based framework [7, 8] and can easily deal with the sequential nature of handwriting. In our approach, we model an object as a set of local parts arranged at specific spatial locations.

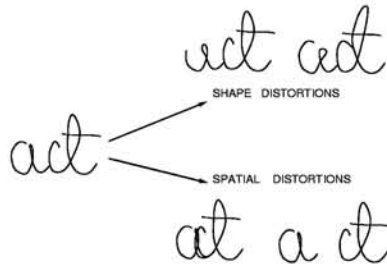

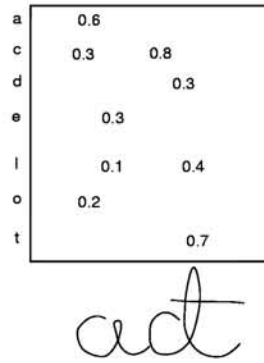

Figure 1: Effect of *shape* distortion, and *spatial* distortions applied on the word "act".

Figure 2: Some of the non-zero elements of the detection matrix associated with the word "act".

Parts-based representation has been used in face detection systems [3] and has recently been applied to spotting keywords in cursive handwriting data [4]. Although the model proposed in [4] presents a rigorous probabilistic approach, it only models the positions of key-points and, in order to learn the appropriate statistics, it requires many ground-truthed training examples.

In this work, we focus on modeling one dimensional objects. In our application, an object is a handwritten word and its parts are the letters. However, the method we propose is quite general and can easily be extended to two dimensional problems.

## 2 The Objective Function

In our approach, we assume that a handwritten pattern is a distorted version of one of the dictionary words. Furthermore, we assume that any distortion of a word can be expressed as a combination of two types of local distortions [6]: a) *shape* distortions of one or more letters, and b) *spatial* distortions, also called *domain warping*, as illustrated in Figure 1. In the latter case, the shape of each letter is unchanged but the location of one or more letters is perturbed.

Shape distortions can be captured using "letter detectors". A number of different techniques can be used to construct letter detectors. In our implementation, we use a neural network-based approach. The output of a letter detector is in the range $[0 - 1]$, where 1 corresponds to the undistorted shape of the corresponding letter.

Since it is not known, a priori, where the letters are located in the pattern, letter detectors, for each letter of the alphabet, are arranged over the pattern so that the pattern is completely covered by their (overlapping) receptive fields. The outputs of the letter detectors form a *detection* matrix, Figure 2. Each row of the *detection* matrix represents one letter and each column corresponds to the position of the letter within the pattern. An element of the detection matrix is labeled as $d^k(x)$, where $k$ denotes the class of the letter, $k \in [1, ..., 26]$, and the $x$ represents the column number. In general, the detection matrix contains a large number of "false alarms" due to the fact that local segments are often ambiguous. The recognition system segments a pattern by selecting one detection matrix element for each letter

of a given dictionary word [1].

To measure spatial distortions, one must first choose a reference point from which distortions are measured. It is clear that for any choice of reference point, the location estimates for letters that are not close to the reference point might be very poor. For this reason, we chose a representation in which each letter serves as a reference point to estimate the position of every other letter. This representation allows translation invariant recognition, is very robust (since it does not depend on any single reference point) and very accurate (since it includes nearest neighbor reference points).

To evaluate the level of distortion of a handwritten pattern from a given dictionary word, we introduce an objective function. The value of this function represents the amount of distortion of the pattern from the dictionary word. We require that the objective function reach a minimal value if all the letters that constitute the dictionary word are detected with the highest confidence and are located at the locations with highest expectation values. Furthermore, we require that the dependence of the function on one of its letters be smaller for longer words.

One function with similar properties to these is the energy function of a system of interacting particles, $\sum_{i,j} q_i U_{i,j}(x_i, x_j) q_j$. If we assume that all the letters are of the same size, we can map 1) letter detection estimates into "charge" and 2) choose interaction terms (potentials) to reflect the expected relative positioning of the letters (detection matrix elements). The energy function of the $n-th$ dictionary word, is then

$$E^n(\vec{x}) = \sum_{i,j=1, i \neq j}^{L_n} d_i^n(x_i) U_{i,j}^n(x_i, x_j) d_j^n(x_j), \qquad (1)$$

where $L_n$ is the number of letters in the word, $x_i$ is the location of the $i-th$ letter of the $n-th$ dictionary word, and $\vec{x} = (x_1, \cdots, x_{L_n})$ is a particular configuration of detection matrix elements. Although this equation has a similar form as, say, the Coulomb energy, it is much more complicated. The interaction terms $U_{i,j}$ are more complex than $1/r$, and each "charge", $d_i(x_i)$, does not have a fixed value, but depends on its location. Note that this energy is a function of a specific choice of elements from the detection matrix, $\vec{x}$, a specific segmentation of the word.

Interaction terms can be calculated from training data in a number of different ways. One possibility is to use the EM algorithm [9] and do the training for each dictionary word. Another possibility is to propagate nearest neighbor estimates. Let us denote with the symbol $p_{ij}^n(x_i, x_j)$ the (pairwise) probability of finding the $j-th$ letter of the $n-th$ dictionary word at distance $x = x_j - x_i$ from the location of the $i-th$ letter. A simple way to approximate pairwise probabilities is to find the probability distribution of letter widths for each letter and then from single letter distributions calculate nearest neighbor pairwise probabilities. Knowing the nearest neighbor probabilities, it is then easy to propagate them and find the pairwise probabilities between any two letters of any dictionary word [7]. Interaction potentials are related to pairwise probabilities (using the Boltzmann distribution and setting $\beta = 1/kT = 1$), as $U_{i,j}^n(x_i, x_j) = -\ln p_{ij}^n(x_i, x_j) + C$.

Since the interaction potentials are defined up to a constant, we can selectively

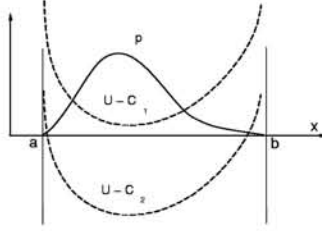 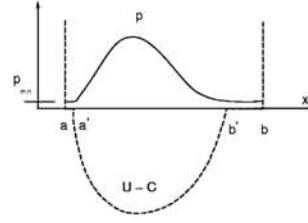

Figure 3: Solid line: an example of a pairwise probability distribution for neighboring letters. Dashed lines: a family of corresponding interaction potentials.

Figure 4: Modified interaction potential. Regions $x \leq a$ and $x \geq b$ are the "forbidden" regions for letter locations. In the regions $a < x < a'$ and $b' < x < b$ the interaction term is zero.

change the value of their minima by choosing different values for $C$, Fig. 3. It is important to stress that the only valid domain for the interaction terms is the region for which $U_{i,j} < 0$ since for each pair of letters $(i,j)$ we want to simultaneously minimize the interaction term $U_{i,j}$ and to maximize the term $d_i \cdot d_j$ [2]. We will assume that there is a value, $p_{min}$, for the pairwise probability below which the estimate of the letter location is not reliable. So, for every $p_{ij}$ such that $0 < p_{ij} < p_{min}$, we set $p_{ij} = p_{min}$. We choose the value of the constant such that $U_{i,j} = -ln(p_{min}) + C = 0$, Fig. 4. In practice, this means that there is an effective range of influence for each letter, and beyond that range the influence of the letter is zero. In the limiting case, one can get a nearest neighbor approximation by appropriately setting $p_{min}$.

It is clear that the interaction terms put constraints on the possible locations of the letters of a given dictionary word. They define "allowed" regions, where the letters can be found, unimportant regions, where the influence of a letter on other letters is zero, and not-allowed regions ($U = \infty$), which have zero probability of finding a letter in that region, Fig. 4.

The task of recognition can now be formulated as follows. For a given dictionary word, find the configuration of elements from the detection matrix (a specific segmentation of the pattern) such that the energy is minimal. Then, in order to find the best dictionary word, repeat the previous procedure for every dictionary word and associate with the pattern the dictionary word with lowest energy. If we denote by $\chi$ the space of all possible segmentations of the pattern, then the final segmentation of the pattern, $\vec{x}^*$, is given as

$$\vec{x}^* = argmin_{\vec{x} \in \chi, n \in N}(E^n(\vec{x})). \tag{2}$$

where the index $n$ runs through the dictionary words.

## 3 Implementation and an Overview of the System

An overview of the system is illustrated in Fig. 5. A raw data file, representing a handwritten word, contains $x$ and $y$ positions of the pen recorded every 10 milliseconds. This input signal is first transformed into *strokes*, which are defined as lines between points with zero velocity in the $y$ direction. Each stroke is characterized by

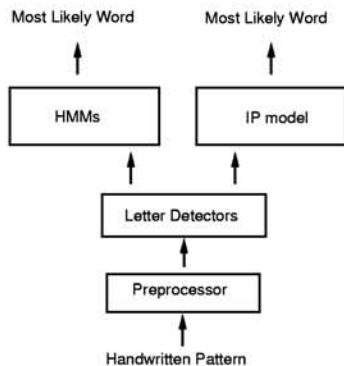

Figure 5: An overview of the system.

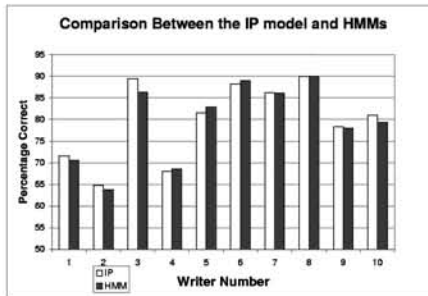

Figure 6: Comparison of recognition results on 10 writers using the IP model and HMMs.

a set of features as suggested in [10]. The preprocessor extracts these features from each stroke and supplies them to the neural network. We have built a multi-layer feedforward network based on a weight sharing technique to detect letters. This particular architecture was proposed by Rumelhart [10]. Similar networks can also be found in literature by the name Time Delay Neural Network, (TDNN) [11]). In our implementation, the network has one hidden layer with thirty rows of hidden units. For details of the network architecture see [10, 7]. The output of the NN, the detection matrix, is then supplied to the HMM-based and IP model-based post-processors, Fig. 5. For both models, we assume that every letter has the same average width.

**Interaction Terms.** The first approximation for interaction terms is to assume a "square well" shape. Each interaction term is then defined with only three parameters, the left boundary $a$, the right boundary $b$ and the depth of the well, $e_n$, which are the same for all the nearest neighbor letters, Fig. 7. The lower and upper limits for the $i-th$ and $j-th$ non-adjacent interaction terms can then be approximated as $a_{ij} = |j - i| \cdot a$ and $b_{ij} = |j - i| \cdot b$, respectively.

**Nearest Neighbor Approximation.** Since the *exact* solution of the energy function given by Eq. (2) is often computationally infeasible (the detection matrices can exceed 40 columns in width for long words), one has to use some approximation technique. One possible solution is suggested in [7], where contextual information is used to constrain the search space. Another possibility is to revise the energy function by considering only nearest neighbor terms and then solve it exactly using a Dynamic Programming (DP) algorithm. We have used DP to find the optimal segmentation for each word. We then use this "optimal" configuration of letters to calculate the energy given by Eq. (1). It is important to mention that we have introduced beginning (B) and end (E) "letters" to mark the beginning and end of the pattern, and their detection probabilities are set to some constant value [3].

**Hidden Markov Models.** The goal of the recognition system is to find the dictionary word with the maximum posterior probability, $p(w|\vec{O}) = p(\vec{O}|w)p(w)/p(\vec{O})$,

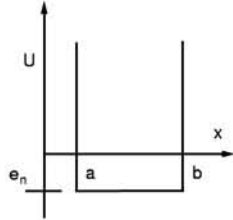
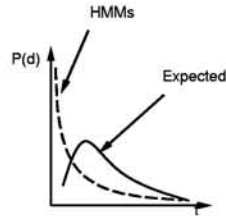

Figure 7: Square well approximation of the interaction potential. Allowed region is defined as $a < x < b$, and forbidden regions are $x < a$, and $x > b$.

Figure 8: The probability of remaining in the same state for exactly $d$ time steps: HMMs (dashed line) vs. expected probability (solid line).

given the handwritten pattern, $\vec{O}$. Since $p(\vec{O})$ and $p(w)$ are the same for all dictionary words, maximizing $p(w|\vec{O})$ is equivalent to maximizing $p(\vec{O}|w)$. To find $p(\vec{O}|w)$, we constructed a left-right (or Bakis) HMM [9] for each dictionary word, $\lambda^n$, where each letter was represented with one state. Given a dictionary word (a model $\lambda^n$), we calculated the maximum likelihood, $p(\vec{O}|\lambda^n) = \sum_{all\ \vec{Q}} P(\vec{O}, \vec{Q}|\lambda^n) = \sum_{all\ \vec{Q}} P(\vec{O}|\vec{Q}, \lambda^n)p(\vec{Q}|\lambda^n)$, where the summation is done over all possible state sequences. We used the *forward-backward procedure* [9] for calculating the previous sum. Emission probabilities were calculated from the detection probabilities using Bayes' rule $P(O_x|q_k) = d^k(x)P(O_x)/P(q_k)$, where $P(q_k)$ denotes the frequency of the $k-th$ letter in the dictionary and the term $P(O_x)$ is the same for all words and can therefore be omitted. Transition probabilities were adjusted until the best recognition results were obtained. Recall that we assumed that all letter widths are the same and therefore the transition probabilities are independent of letter pairs.

## 4    Results and Discussion

Our dataset (obtained from David Rumelhart [10]) consists of words written by 100 different writers, where each writer wrote 1000 words. The size of the dictionary is 1000 words. The neural network was trained on 70 writers (70,000 words) and an independent group of writers was used as a cross validation set. We have tested both the IP model and HMMs on a group of 10 writers (different from the testing and cross-validation groups). The results for each model are depicted in Fig. 6. The IP model chose the correct word 79.89% of the time, while HMMs selected the correct word 79.44% of the time. Although the overall performance of the two models was almost identical, the results differ by several percent on individual writers. This suggests that our model could be used in combination with HMMs (e.g. with some averaging technique) to improve overall recognition.

It is important to mention that new dictionary words can be easily added to the dictionary and the IP model does not require retraining on the new words (using the method of calculating interaction terms suggested in this paper). The only information about the new word that has to be supplied to the system is the ordering of the letters. Knowing the nearest neighbor pairwise probabilities, $p_{ij}^n(x_i, x_j)$, it is easy to calculate the location estimates between *any* two letters of the new word. Furthermore, the IP model can easily recognize words where many of the letters are

highly distorted or missing.

In standard first-order HMMs with time-independent transition probabilities, the probability of remaining in the $i - th$ state for exactly $d$ time steps is illustrated in Fig. 8. The real probability distribution on letter widths is actually similar to a Poisson distribution [11], Fig. 8. It has been shown that explicit duration HMMs can significantly improve recognition accuracy, but at the expense of a significant increase in computational complexity [5]. Our model, on the other hand, can easily model arbitrarily complex pairwise probabilities without increasing the computational complexity (using DP in a nearest neighbor approximation). We think that this is one of the biggest advantages of our approach over HMMs. We believe that including more precise interaction terms will yield significantly better results (as in HMMs) and this work is currently in progress.

## Acknowledgments

Supported in part by the Office of Naval Research. The authors thank the members of Institute for Brain and Neural Systems for helpful conversations.

## Footnotes

\*Now at MIT Lincoln Laboratory, Lexington, MA 02420-9108

[1]Note that this segmentation corresponds to finding the centers of the letters, as opposed to segmenting a word into letters by finding their boundaries.

[2] For $U_{i,j} > 0$, increasing $d_i \cdot d_j$ would increase, rather than decrease, the energy function.

[3]This is necessary in order to define interaction potentials for single letter words.

## References

[1] Y. Bengio, Y. LeCun, C. Nohl, and C. Burges. Lerec: A NN/HMM hybrid for on-line handwriting recognition. *Neural Computation*, 7:1289–1303, 1995.

[2] H. Bourlard and C. Wellekens. Links between hidden Markov models and multilayer perceptrons. *IEEE Transactions on PAMI*, 12:1167–1178, 1990.

[3] M. Burl, T. Leung, and P. Perona. Recognition of planar object classes. In *Proc. IEEE Comput. Soc. Conf. Comput. Vision and Pattern Recogn.*, 1996.

[4] M. Burl and P. Perona. Using hierarchical shape models to spot keywords in cursive handwriting data. In *Proc. CVPR 98*, 1998.

[5] C. Mitchell and L. Jamieson. Modeling duration in a hidden markov model with the exponential family. In *Proc. ICASSP*, pages 331–334, 1993.

[6] D. Mumford. Neuronal archetectures for pattern theoretic problems. In K. C. and D. J. L., editors, *Large-Scale Neuronal theories of the Brain*, pages 125–152. MIT Press, Cambridge, MA, 1994.

[7] P. Neskovic. *Feedforward, Feedback Neural Networks With Context Driven Segmentation And Recognition*. PhD thesis, Brown University, Physics Dept., May 1999.

[8] P. Neskovic and L. Cooper. Neural network-based context driven recognition of on-line cursive script. In *7th IWFHR*, 2000.

[9] L. Rabiner and B. Juang. An introduction to hidden markov models. *ASSP magazine*, 3(1):4–16, 1986.

[10] D. E. Rumelhart. Theory to practice: A case study – recognizing cursive handwriting. In E. B. Baum, editor, *Computational Learning and Cognition: Proceedings of the Third NEC Research Symposium*. SIAM, Philadelphia, 1993.

[11] M. Schenkel, I. Guyon, and D. Henderson. On-line cursive script recognition using time delay neural networks and hidden markov models. *Machine Vision and Applications*, 8:215–223, 1995.
